# The Maximal Causes of Natural Scenes are Edge Filters

**Gervasio Puertas**∗
Frankfurt Institute for Advanced Studies
Goethe-University Frankfurt, Germany
puertas@fias.uni-frankfurt.de

**Jörg Bornschein**∗
Frankfurt Institute for Advanced Studies
Goethe-University Frankfurt, Germany
bornschein@fias.uni-frankfurt.de

**Jörg Lücke**
Frankfurt Institute for Advanced Studies
Goethe-University Frankfurt, Germany
luecke@fias.uni-frankfurt.de

## Abstract

We study the application of a strongly non-linear generative model to image patches. As in standard approaches such as Sparse Coding or Independent Component Analysis, the model assumes a sparse prior with independent hidden variables. However, in the place where standard approaches use the sum to combine basis functions we use the maximum. To derive tractable approximations for parameter estimation we apply a novel approach based on variational Expectation Maximization. The derived learning algorithm can be applied to large-scale problems with hundreds of observed and hidden variables. Furthermore, we can infer all model parameters including observation noise and the degree of sparseness. In applications to image patches we find that Gabor-like basis functions are obtained. Gabor-like functions are thus not a feature exclusive to approaches assuming linear superposition. Quantitatively, the inferred basis functions show a large diversity of shapes with many strongly elongated and many circular symmetric functions. The distribution of basis function shapes reflects properties of simple cell receptive fields that are not reproduced by standard linear approaches. In the study of natural image statistics, the implications of using different superposition assumptions have so far not been investigated systematically because models with strong non-linearities have been found analytically and computationally challenging. The presented algorithm represents the first large-scale application of such an approach.

## 1  Introduction

If Sparse Coding (SC, [1]) or Independent Component Analysis (ICA; [2, 3]) are applied to image patches, basis functions are inferred that closely resemble Gabor wavelet functions. Because of the similarity of these functions to simple-cell receptive fields in primary visual cortex, SC and ICA became the standard models to explain simple-cell responses, and they are the primary choice in modelling the local statistics of natural images. Since they were first introduced, many different versions of SC and ICA have been investigated. While many studies focused on different ways to efficiently infer the model parameters (e.g. [4, 5, 6]), many others investigated the assumptions used in the underlying generative model itself. The modelling of observation noise can thus be regarded as the major difference between SC and ICA (see, e.g., [7]). Furthermore, different forms of independent sparse priors have been investigated by many modelers [8, 9, 10], while other approaches have gone a step further and studied a relaxation of the assumption of independence between hidden variables [11, 12, 13].

---

∗authors contributed equally

An assumption that has, in the context of image statistics, been investigated relatively little is the assumption of linear superpositions of basis functions. This assumption is not only a hallmark of SC and ICA but, indeed, is an essential part of many standard algorithms including Principal Component Analysis (PCA), Factor Analysis (FA; [14]), or Non-negative Matrix Factorization (NMF; [15]). For many types of data, linear superposition can be motivated by the actual combination rule of the data components (e.g., sound waveforms combine linearly). For other types of data, including visual data, linear superposition can represent a severe approximation, however. Models assuming linearity are, nevertheless, often used because they are easier to study analytically and many derived algorithms can be applied to large-scale problems. Furthermore, they perform well in many applications and may, to certain extents, succeed well in modelling the distribution, e.g., of local image structure. From the perspective of probabilistic generative models, a major aim is, however, to recover the actual data generating process, i.e., to recover the actual generating causes (see, e.g., [7]). To accomplish this, the crucial properties of the data generation should be modelled as realistically as possible. If the data components combine non-linearly, this should thus be reflected by the generative model. Unfortunately, inferring the parameters in probabilistic models assuming non-linear superpositions has been found to be much more challenging than in the linear case (e.g. [16, 17, 18, 19], also compare [20, 21]). To model image patches, for instance, large-scale applications of non-linear models, with the required large numbers of observed and hidden variables, have so far not been reported.

In this paper we study the application of a probabilistic generative model with strongly non-linear superposition to natural image patches. The basic model has first been suggested in [19] where tractable learning algorithms for parameter optimization where inferred for the case of a superposition based on a point-wise maximum. The model (which was termed *Maximal Causes Analysis*; MCA) used a sparse prior for independent and binary hidden variables. The derived algorithms compared favorably with state-of-the-art approaches on standard non-linear benchmarks and they were applied to realistic data. However, the still demanding computational costs limited the application domain to relatively small-scale problems. The unconstrained model for instance was used with at most $H = 20$ hidden units. Here we use a novel learning algorithm to infer the parameters of a variant of the MCA generative model. The approach allows for scaling the model up to several hundreds of observed and hidden variables. It enables large-scale applications to image patches and, thus, allows for studying the inferred basis functions as it is commonly done for linear approaches.

## 2 The Maximal Causes Generative Model

Consider a set of $N$ data points $\{\vec{y}^{(n)}\}_{n=1,...,N}$ sampled independently from an underlying distribution ($\vec{y}^{(n)} \in \mathbb{R}^{D \times 1}$, $D$ is the number of observed variables). For these data we seek parameters $\Theta = (W, \sigma, \pi)$ that maximize the data likelihood $\mathcal{L} = \prod_{n=1}^{N} p(\vec{y}^{(n)} | \Theta)$ under a variant of the MCA generative model [19] which is given by:

$$p(\vec{s} | \Theta) \quad = \quad \prod_h \pi^{s_h} (1 - \pi)^{1-s_h}, \qquad \text{(Bernoulli distribution)} \tag{1}$$

$$p(\vec{y} | \vec{s}, \Theta) \quad = \quad \prod_d \mathcal{N}(y_d; \overline{W}_d(\vec{s}, W), \sigma^2), \quad \text{where } \overline{W}_d(\vec{s}, W) = \max_h \{s_h W_{dh}\} \tag{2}$$

and where $\mathcal{N}(y_d; w, \sigma^2)$ denotes a scalar Gaussian distribution. $H$ denotes the number of hidden variables $s_h$, and $W \in \mathbb{R}^{D \times H}$. The model differs from the one previously introduced by the use of Gaussian noise instead of Poisson noise in [19]. Eqn. 2 results in the basis functions $\vec{W}_h = (W_{1h}, \ldots, W_{Dh})^T$ of the MCA model to be combined non-linearly by a point-wise maximum. This becomes salient if we compare (2) with the linear case using the vectorial notation $\max_h\{\vec{W}_h'\} = (\max_h\{W_{1h}'\}, \ldots, \max_h\{W_{Dh}'\})^T$ for vectors $\vec{W}_h' \in \mathbb{R}^{D \times 1}$:

$$p(\vec{y} | \vec{s}, \Theta) \quad = \quad \mathcal{N}(\vec{y}; \max_h\{s_h \vec{W}_h\}, \sigma^2 \, \mathbb{1}) \qquad \text{(non-linear superposition)} \tag{3}$$

$$p(\vec{y} | \vec{s}, \Theta) \quad = \quad \mathcal{N}(\vec{y}; \sum_h s_h \vec{W}_h, \sigma^2 \, \mathbb{1}) \qquad \text{(linear superposition)} \tag{4}$$

where $\mathcal{N}(\vec{y}; \vec{\mu}, \Sigma)$ denotes the multi-variate Gaussian distribution (note that $\sum_h s_h \vec{W}_h = W\vec{s}$). As in linear approaches such as SC, the combined basis functions set the mean values of the observed variables $y_d$, which are independently and identically drawn from Gaussian distributions

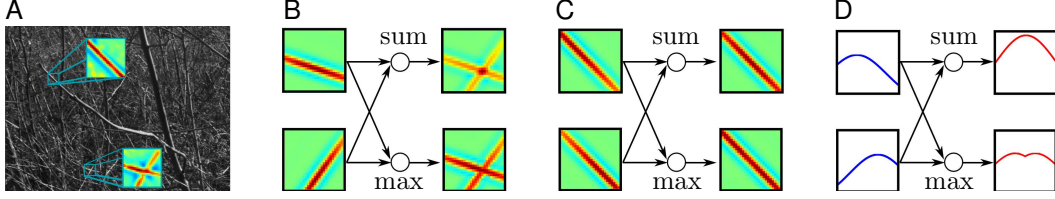

Figure 1: **A** Example patches extracted from an image and preprocessed using a Difference of Gaussians filter. **B** Two generated patches constructed from two Gabor basis functions with approximately orthogonal wave vectors. In the upper-right the basis functions were combined using linear superposition. In the lower-right they were combined using a point-wise maximum (note that the max was taken after channel-splitting (see Eqn. 15 and Fig. 2). **C** Superposition of two collinear Gabor functions using the sum (upper-right) or point-wise maximum (lower-right). **D** Cross-sections through basis functions (along maximum amplitude direction). Left: Cross-sections through two different collinear Gabor functions (compare C). Right: Cross-sections through their superpositions using sum (top) and max (bottom).

with variance $\sigma^2$ (Eqn. 2). The difference between linear and non-linear superposition is illustrated in Fig. 1. In general, the maximum superposition results in much weaker interferences. This is the case for diagonally overlapping basis functions (Fig. 1B) and, at closer inspection, also for overlapping collinear basis functions (Fig. 1C,D). Strong interferences as with linear combinations can not be expected from combinations of image components. For preprocessed image patches (compare Fig. 4D), it could thus be argued that the maximum combination is closer to the actual combination rule of image causes. In any case, the maximum represents an alternative to study the implications of combination rules in the image domain.

To optimize the parameters $\Theta$ of the MCA model (1) and (2), we use a variational EM approach (see, e.g., [22]). That is, instead of maximizing the likelihood directly, we maximize the free-energy:

$$\mathcal{F}(q,\Theta)=\sum_{n=1}^{N}\left[\sum_{\vec{s}} q^{(n)}(\vec{s};\Theta')\left[\log\left(p(\vec{y}^{(n)}\,|\,\vec{s},W,\sigma)\right)+\log\left(p(\vec{s}\,|\,\pi)\right)\right]\right]+H(q)\,,\quad(5)$$

where $q^{(n)}(\vec{s};\Theta')$ is an approximation to the exact posterior. In the variational EM scheme $\mathcal{F}(q,\Theta)$ is maximized alternately with respect to $q$ in the E-step (while $\Theta$ is kept fixed) and with respect to $\Theta$ in the M-step (while $q$ is kept fixed). As a multiple-cause model, an exact E-step is computationally intractable for MCA. Additionally, the M-step is analytically intractable because of the non-linearity in MCA. The computational intractability in the E-step takes the form of expectation values of functions $g$, $\langle g(\vec{s})\rangle_{q^{(n)}}$. These expectations are intractable if the optimal choice of $q^{(n)}$ in (5) is used (i.e., if $q^{(n)}$ is equal to the posterior: $q^{(n)}(\vec{s};\Theta')=p(\vec{s}\,|\,\vec{y}^{(n)},\Theta')$). To derive an efficient learning algorithm, our approach approximates the intractable expectations $\langle g(\vec{s})\rangle_{q^{(n)}}$ by truncating the sums over the hidden space of $\vec{s}$:

$$\langle g(\vec{s})\rangle_{q^{(n)}}\;=\;\frac{\sum_{\vec{s}}p(\vec{s},\vec{y}^{(n)}\,|\,\Theta')\,g(\vec{s})}{\sum_{\widetilde{\vec{s}}}p(\widetilde{\vec{s}},\vec{y}^{(n)}\,|\,\Theta')}\;\approx\;\frac{\sum_{\vec{s}\in\mathcal{K}_n}p(\vec{s},\vec{y}^{(n)}\,|\,\Theta')\,g(\vec{s})}{\sum_{\widetilde{\vec{s}}\in\mathcal{K}_n}p(\widetilde{\vec{s}},\vec{y}^{(n)}\,|\,\Theta')}\,,\quad(6)$$

where $\mathcal{K}_n$ is a small subset of the hidden space. Eqn. 6 represents a good approximation if the set $\mathcal{K}_n$ contains most of the posterior probability mass. The approximation will be referred to as *Expectation Truncation* and can be derived as a variational EM approach (see Suppl. A). For other generative models similar truncation approaches have successfully been used [19, 23]. For the learning algorithm, $\mathcal{K}_n$ in (6) is chosen to contain hidden states $\vec{s}$ with at most $\gamma$ active causes $\sum_h s_h \le \gamma$. Furthermore, we only consider the combinatorics of $H' \ge \gamma$ hidden variables. More formally we define:

$$\mathcal{K}_n\;=\;\{\vec{s}\,|\,\left(\textstyle\sum_j s_j \le \gamma \text{ and } \forall i \notin I : s_i = 0\right) \text{ or } \textstyle\sum_j s_j \le 1\},\quad(7)$$

where the index set $I$ contains those $H'$ hidden variables that are the most likely to have generated data point $\vec{y}^{(n)}$ (the last term in Eqn. 7 assures that all states $\vec{s}$ with just one non-zero entry are also

evaluated). To determine the $H'$ hidden variables for $I$ we use those variables $h$ with the $H'$ largest values of a *selection function* $\mathcal{S}_h(\vec{y}^{(n)})$ which is given by:

$$\mathcal{S}_h(\vec{y}^{(n)}) \;=\; \pi\, \mathcal{N}(\vec{y}^{(n)};\, \vec{W}_h^{\text{eff}}, \sigma^2\, \mathbb{1})\,, \quad \text{with an effective weight} \quad W_{dh}^{\text{eff}} = \max\{y_d, W_{dh}\}\,. \tag{8}$$

Selecting hidden variables based on $\mathcal{S}_h(\vec{y}^{(n)})$ is equivalent to selecting them based on an upper bound of $p(s_h{=}1 \,|\, \vec{y}^{(n)}, \Theta)$. To see this note that $p(\vec{y}^{(n)} \,|\, \Theta)$ is independent of $h$ and that:

$$\frac{p(s_h{=}1 \,|\, \vec{y}^{(n)}, \Theta)}{p(\vec{y}^{(n)} \,|\, \Theta)} = \sum_{\substack{\vec{s} \\ s_h = 1}} \big( \prod_d p(y_d^{(n)} \,|\, \overline{W}_d(\vec{s}, W), \sigma) \big) p(\vec{s} \,|\, \pi) \le \big( \prod_d p(y_d^{(n)} \,|\, W_{dh}^{\text{eff}}, \sigma) \big) \sum_{\substack{\vec{s} \\ s_h = 1}} p(\vec{s} \,|\, \pi),$$

with the right-hand-side being equal to $\mathcal{S}_h(\vec{y}^{(n)})$ in Eqn. 8 (see Suppl. B for details). A low value of $\mathcal{S}_h(\vec{y}^{(n)})$ thus implies a low value of $p(s_h = 1 \,|\, \vec{y}^{(n)}, \Theta)$ and hence a low likelihood that cause $h$ has generated data point $\vec{y}^{(n)}$. In numerical experiments on ground-truth data we have verified that for most data points Eqn. 6 with Eqn. 7 indeed finally approximates the true expectation values with high accuracy.

Having derived tractable approximations for the expectation values (6) in the E-step, let us now derive parameter update equations in the M-step. An update rule for the weight matrix $W$ of this model was derived in [19] and is given by:

$$W_{dh}^{\text{new}} = \frac{\displaystyle\sum_{n \in \mathcal{M}} \langle \mathcal{A}_{dh}^{\rho}(\vec{s}, W) \rangle_{q^{(n)}} y_d^{(n)}}{\displaystyle\sum_{n \in \mathcal{M}} \langle \mathcal{A}_{dh}^{\rho}(\vec{s}, W) \rangle_{q^{(n)}}}, \qquad \mathcal{A}_{dh}^{\rho}(\vec{s}, W) = \left( \frac{\partial}{\partial W_{dh}} \overline{W}_d^{\rho}(\vec{s}, W) \right), \tag{9}$$

$$\overline{W}_d^{\rho}(\vec{s}, W) = \left( \sum_{h=1}^{H} (s_h W_{dh})^{\rho} \right)^{\frac{1}{\rho}}, \tag{10}$$

where the parameter $\rho$ is set to a large value (we used $\rho = 20$). The derivation of the update rule for $\sigma$ (Gaussian noise has previously not been used) is straight-forward, and the update equation is given by:

$$\sigma^{\text{new}} = \sqrt{ \frac{1}{|\mathcal{M}|\, D} \sum_{n \in \mathcal{M}} \big\langle \big|\big| \vec{y}^{(n)} - \max_h\{s_h\, \vec{W}_h\} \big|\big|^2 \big\rangle_{q_n} }\,. \tag{11}$$

Note that in (9) to (11) we do not sum over all data points $\vec{y}^{(n)}$ but only over those in a subset $\mathcal{M}$ ($|\mathcal{M}|$ is the number of elements in $\mathcal{M}$). The subset contains the data points for which (6) finally represents a good approximation. It is defined to contain the $N^{\text{cut}}$ data points with largest values $\sum_{\vec{s} \in \mathcal{K}_n} p(\vec{s}, \vec{y}^{(n)} \,|\, \Theta')$, i.e., with the largest values for the denominator in (6). $N^{\text{cut}}$ is hereby the expected number of data points that have been generated by states with less or equal to $\gamma$ non-zero entries:

$$N^{\text{cut}} = N \sum_{\vec{s},\, |\vec{s}| \le \gamma} p(\vec{s} \,|\, \pi) = N \sum_{\gamma'=0}^{\gamma} \binom{H}{\gamma'} \pi^{\gamma'} (1 - \pi)^{H - \gamma'}\,. \tag{12}$$

The selection of data points is an important difference to earlier truncation approaches (compare [19, 23]), and its necessity can be shown analytically (Suppl. A).

Update equations (9), (10), and (11) have been derived by setting the derivatives of the free-energy (w.r.t. $W$ and $\sigma$) to zero. Similarly, we can derive the update equation for the sparseness parameter $\pi$. However, as the approximation only considers states $\vec{s}$ with a maximum of $\gamma$ non-zero entries, the update has to correct for an underestimation of $\pi$ (compare Suppl. A). If such a correction is taken into account, we obtain the update rule:

$$\pi^{\text{new}} = \frac{A(\pi)\, \pi}{B(\pi)} \frac{1}{|\mathcal{M}|} \sum_{n \in \mathcal{M}} \langle |\vec{s}| \rangle_{q_n} \quad \text{with } |\vec{s}| = \sum_{h=1}^{H} s_h\,, \tag{13}$$

$$A(\pi) = \sum_{\gamma'=0}^{\gamma} \binom{H}{\gamma'} \pi^{\gamma'} (1 - \pi)^{H - \gamma'} \text{ and } B(\pi) = \sum_{\gamma'=0}^{\gamma} \gamma' \binom{H}{\gamma'} \pi^{\gamma'} (1 - \pi)^{H - \gamma'}\,. \tag{14}$$

Note that the correction factor $\frac{A(\pi)\, \pi}{B(\pi)}$ in (13) is equal to one over $H$ if we allow for all possible states (i.e., $\gamma = H' = H$). Also the set $\mathcal{M}$ becomes equal to the set of all data points in this case (because

$N^{\text{cut}} = N$). For $\gamma = H' = H$, Eqn. 13 thus falls back to the exact EM update rule that can canonically be derived by setting the derivative of (5) w.r.t. $\pi$ to zero (while using the exact posterior). Also the update equations (9), (10), and (11) fall back to their canonical form for $\gamma = H' = H$. By choosing a $\gamma$ between one and $H$ we can thus choose the accuracy of the used approximation. The higher the value of $\gamma$ the more accurate is the approximation but the larger are also the computational costs. For intermediate values of $\gamma$ we can obtain very good approximations with small computational costs. Crucial for the scalability to large-scale problems is hereby the preselection of $H' < H$ hidden variables using the selection function in Eqn. 8.

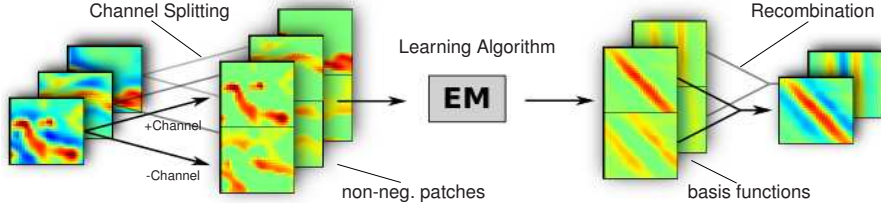

Figure 2: Illustration of patch preprocessing and basis function visualization. The left-hand-side shows data points obtained from gray-value patches after DoG filtering. These patches are transformed to non-negative data by Eqn. 15. The algorithm maximizes the data likelihood under the MCA model (1) and (2), and infers basis functions (second from the right). For visualization, the basis functions are displayed after their parts have been recombined again.

## 3 Numerical Experiments

The update equations (9), (10), (11), and (13) together with approximation (6) with (7) and (8) define a learning algorithm that optimizes the full set of parameters of the MCA generative model (1) and (2). We will apply the algorithm to visual data as received by the primary visual cortex of mammals. In mammals, visual information is transferred to the cortex via two types of neurons in the lateral geniculus nucleus (LGN): center-on and center-off cells. The sensitivity of center-on neurons can be modeled by a Difference of Gaussians (DoG) filter with positive central part, while the sensitivity of center-off cells can be modelled by an inverted such filter. A model for preprocessing of an image patch is thus given by a DoG filter and a successive splitting of the positive and the negative parts of the filtered image. More formally, we use a DoG filter to generate patches $\tilde{\vec{y}}$ with $\tilde{D} = 26 \times 26$ pixels. Such a patch is then converted to a patch of size $D = 2\tilde{D}$ by assigning:

$$y_d = [\tilde{y}_d]^+ \quad \text{and} \quad y_{D+d} = [-\tilde{y}_d]^+ \qquad (15)$$

(for $d = 1, \ldots, D$) where $[x]^+ = x$ for $x \geq 0$ and $[x]^+ = 0$ otherwise. This procedure has repeatedly been used in the context of visual data processing (see, e.g., [24]) and is, as discussed, closely aligned with mammalian visual preprocessing (see Fig. 2 for an illustration).

Before we applied the algorithm to natural image patches, it was first evaluated on artificial data with ground-truth. As inferred basis functions of images most commonly resemble Gabor wavelets, we used Gabor functions for the generation of artificial data. The Gabor basis functions were combined according to the MCA generative model (1) and (2). We used $H^{\text{gen}} = 400$ Gabor functions for generation. The variances of the Gaussian envelop of each Gabor were sampled from a distribution in $n_x/n_y$-space (Fig. 3C) with $\sigma_x$ and $\sigma_y$ denoting the standard deviations of the Gaussian envelope, and with $f$ denoting the Gabor frequency. Angular phases and centers of the Gabors were sampled from uniform distributions. The wave vector's module was set to 1 ($f = \frac{1}{2\pi}$) and the envelope amplitude was 10. The parameters were chosen to lie in the same range as the parameters inferred in preliminary runs of the algorithm on natural image patches.

For the generation of each artificial patch we drew a binary vector $\vec{s}$ according to (1) with $\pi H^{\text{gen}} = 2$. We then selected the $|\vec{s}|$ corresponding Gabor functions and used channel-splitting (15) to convert them into basis functions with only non-negative parts. To form an artificial patch, these basis functions were combined using the point-wise maximum according to (2). We generated $N = 150\,000$ patches as data points in this way (Fig. 3A shows some examples).

The algorithm was applied with $H = 300$ hidden variables and approximation parameters $\gamma = 3$ and $H' = 8$. We generated the data with a larger number of basis functions to better match the continuous distribution of the real generating components of images. The basis functions $\vec{W}_h$ were initialized

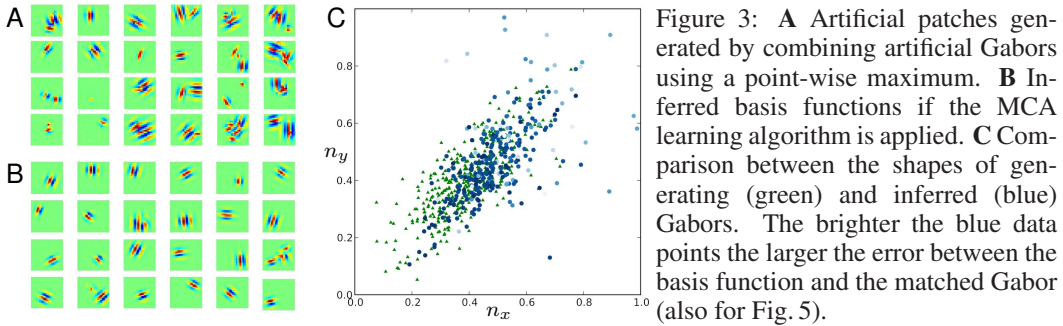

Figure 3: **A** Artificial patches generated by combining artificial Gabors using a point-wise maximum. **B** Inferred basis functions if the MCA learning algorithm is applied. **C** Comparison between the shapes of generating (green) and inferred (blue) Gabors. The brighter the blue data points the larger the error between the basis function and the matched Gabor (also for Fig. 5).

by setting them to the average over all the preprocessed input patches plus a small Gaussian white noise ($\approx 0.5\%$ of the corresponding mean). The initial noise parameter $\sigma$ was set following Eqn. 11 by using all data points (setting $|\mathcal{M}| = N$ initially). Finally, the initial sparseness level was set to $\pi H = 2$. The model parameters were updated according to Eqns. 9 to 13 using 60 EM iterations. To help avoiding local optima, a small amount of Gaussian white noise ($\approx 0.5\%$ of the average basis function value) was added during the first 20 iterations, was linearly decreased to zero between iterations 20 and 40, and kept at zero for the last 20 iterations. During the first 20 iterations the updates considered all $N$ data points ($|\mathcal{M}| = N$). Between iteration number 20 and 40 the amount of used data points was linearly decreased to ($|\mathcal{M}| = N^{\text{cut}}$) where it was kept constant for the last 20 iterations. Considering all data points for the updates initially, has proven beneficial because the selection of data points is based on very incomplete knowledge during the first iterations.

Fig. 3B displays some of the typical basis functions that were recovered in a run of the algorithm on artificial patches. As can be observed (and as could have been expected), they resemble Gabor functions. When we matched the obtained basis functions with Gabor functions (compare, e.g., [25, 26, 27] for details), the Gabor parameters obtained can be analyzed further. We thus plotted the values parameterizing the Gabor shapes in an $n_x/n_y$-plot. This also allowed us to investigate how well the generating distribution of artificial Gabors was recovered. Fig. 3C shows the generating (green) and the recovered distribution of Gabors (blue). Although some few recovered basis functions lie a relatively distant from the generating distribution, it is in general recovered well. The recovered sparseness level was with $\pi H = 2.62$ a bit larger than the initial level of $\pi H^{\text{gen}} = 2$. This is presumably due to the smaller number of basis function in the model $H < H^{\text{gen}}$. Also the finite inferred noise level of $\sigma = 0.37$ (despite a generation without noise) can be explained by this mismatch. Depending on the parameters of the controls, we can observe different amounts of outliers (usually not more than $5\% - 10\%$). These outliers are usually basis functions that represent more than one Gabor or small Gabor parts. Importantly, however, we found that the large majority of inferred Gabors consistently recovered the generating Gabor functions in $n_x/n_y$-plots. In particular, when we changed the angle of the generating distribution in the $n_x/n_y$-plots (e.g., to $25^o$ or $65^o$), the angle of the recovered distributions changed accordingly. Note that these controls are a quantitative version of the artificial Gabor and grating data used for controls in [1].

**Application to Image Patches.** The dataset used in the experiment on natural images was prepared by sampling $N = 200\,000$ patches of $\tilde{D} = 26 \times 26$ pixels from the van Hateren image database [28] (while constraining random selection to patches of images without man-made structures). We preprocessed the patches as described above using a DoG filter[1] with a ratio of $3 : 1$ between positive and negative parts (see, e.g., [29]) before converting the patches using Eqn. 15.

The algorithm was applied with $H = 400$ hidden variables and approximation parameters $\gamma = 4$ and $H' = 12$. We used parameter initialization as described above and ran 120 EM iterations (also as described above). After learning the inferred sparseness level was $\pi H = 1.63$ and the inferred noise level was $\sigma = 1.59$. The inferred basis functions we found to resembled Gabor-like functions at different locations, and with different orientations and frequencies. Additionally, we obtained many globular basis functions with no or very little orientation preferences. Fig. 4 shows a selection of the $H = 400$ functions after a run of the algorithm (see suppl. Fig. C.1 for

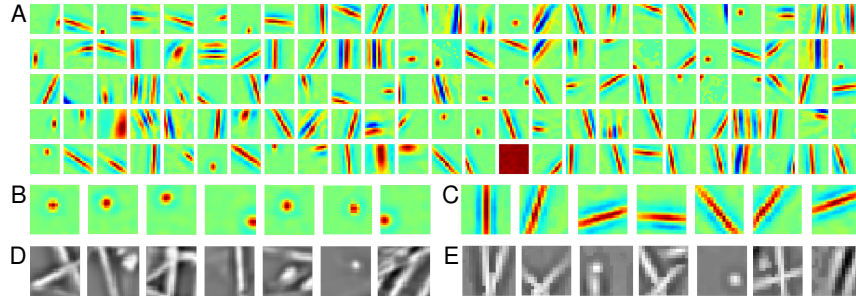

Figure 4: Numerical experiment on image patches. **A** Random selection of 125 basis functions of the $H$=400 inferred. **B** Selection of most globular functions and **C** most elongated functions. **D** Selection of preprocessed patches extracted from natural images. **E** Selection of data points generated according to the model using the inferred basis functions and sparseness level (but no noise).

all functions). The patches in Fig. 4D,E were chosen to demonstrate the high similarity between preprocessed natural patches (in D) and generated ones (in E). To highlight the diversity of obtained basis functions, Figs. 4B,C display some of the most globular and elongated examples, respectively. The variety of Gabor shapes is currently actively discussed [30, 31, 10, 32, 27] since it became obvious that standard linear models (e.g., SC and ICA), could not explain this diversity [33]. To facilitate comparison with earlier approaches, we have applied Gabor matching (compare [25]) and analyzed the obtained parameters. Instead of matching the basis functions directly, we first computed estimates of their corresponding receptive fields (RFs). These estimates were obtained by convoluting the basis functions with the same DoG filter as used for preprocessing (see, e.g., [27] and Suppl. C.1 for details). In controls we found that these convoluted fields were closely matched by RFs estimated using reverse correlation as described, e.g., in [7].

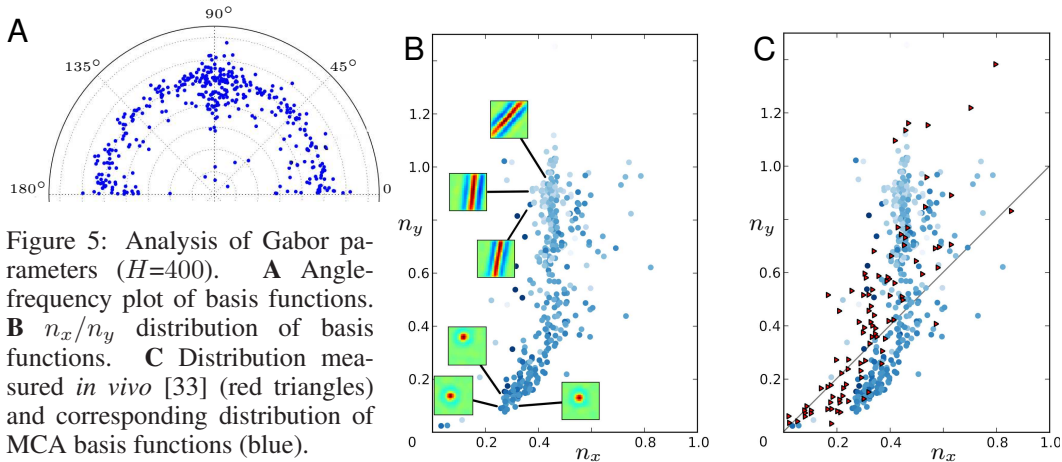

Figure 5: Analysis of Gabor parameters ($H$=400). **A** Angle-frequency plot of basis functions. **B** $n_x/n_y$ distribution of basis functions. **C** Distribution measured *in vivo* [33] (red triangles) and corresponding distribution of MCA basis functions (blue).

After matching the (convoluted) fields with Gabor functions, we found a relatively homogeneous distribution of the fields' orientations as it is commonly observed (Fig. 5A). The frequencies are distributed around 0.1 cycles per pixel, which reflects the band-pass property of the DoG filter. To analyze the Gabor shapes, we plotted the parameters using an $n_x/n_y$-plot (as suggested in [33]). The broad distribution in $n_x/n_y$-space hereby reflects the high diversity of basis functions obtained by our algorithm (see Fig. 5B). The specific form of the obtained shape distribution is, hereby, similar to the distribution of macaque V1 simple cells as measure in *in vivo* recordings [33]. However, the MCA basis functions do quantitatively not match the measurements exactly (see Fig. 5C): the MCA distribution contains a higher percentage of strongly elongated basis functions, and many MCA functions are shifted slightly to the right relative to the measurements. If the basis functions are matched with Gabors directly, we actually do not observe the latter effect (see suppl. Fig. C.2). If simple-cell responses are associated with the posterior probabilities of multiple-cause models, the basis functions should, however, not be compared to measured RFs directly (although it is frequently done in the literature).

To investigate the implications of different numbers of hidden variables, we also ran the algorithm with $H = 200$ and $H = 800$. In both cases we observed qualitatively and quantitatively similar distributions of basis functions. Runs with $H = 200$ thus also contained many circular symmetric basis functions (see suppl. Fig. C.3 for the distribution of shapes). This observation is remarkable because it shows that such 'globular' fields are a very stable feature for the MCA approach, also for small numbers of hidden variables. Based on standard generative models with linear superposition it has recently been argued [32] that such functions are only obtained in a regime with large numbers of hidden variables relative to the input dimensionality (see [34] for an early contribution).

## 4  Discussion

We have studied the application of a strongly non-linear generative model to image patches. The model combines basis functions using a point-wise maximum as an alternative to the linear combination as assumed by Sparse Coding, ICA, and most other approaches. Our results suggest that changing the component combination rule has a strong impact on the distribution of inferred basis functions. While we still obtain Gabor-like functions, we robustly observe a large variety of basis functions. Most notably, we obtain circular symmetric functions as well as many elongated functions that are closely associated with edges traversing the entire patch (compare Figs. 1 and 4). Approaches using linear component combination, e.g. ICA or SC, do usually not show these features. The differences in basis function shapes between non-linear and linear approaches are, in this respect, consistent with the different types of interferences between basis functions. The maximum results in basis function combinations with much less pronounced interferences, while the stronger interferences of linear combinations might result in a repulsive effect fostering less elongated fields (compare Fig. 1).

For linear approaches, a large diversity of Gabor shapes (including circular symmetric fields) could only be obtained in very over-complete settings [34], or specifically modelled priors with hand-set sparseness levels [10]. Such studies were motivated by a recently observed discrepancy of receptive fields as predicted by SC or ICA, and receptive fields as measured *in vivo* [33]. Compared to these measurements, the MCA basis functions and their approximate receptive fields show a similar diversity of shapes. MCA functions and measured RFs both show circular symmetric fields and in both cases there is a tendency towards fields elongated orthogonal to the wave-vector direction (compare Fig. 4). Possible factors that can influence the distributions of basis functions, for MCA as well as for other methods, are hereby different types of preprocessing, different prior distributions, and different noise models. Even if the prior type is fixed, differences for the basis functions have been reported for different settings of prior parameters (e.g., [10]). If possible, these parameters should thus be learned along with the basis functions. All the different factors named above may result in quantitative differences, and the shift of the MCA functions relative to the measurements might have been caused by one of these factors. For the MCA model, possible effects of assuming binary hidden variables remain to be investigated. Presumably, also dependencies between hidden variables as investigated in recent contributions [e.g. 13, 12, 11] play an important role, e.g., if larger structures of specific arrangements of edges and textures are considered. As the components in such models are combined less randomly, the implications of their combination rule may even be more pronounced in these cases.

In conclusion, probably neither the linear nor the maximum combination rule does represent the exact model for local visual component combinations. However, while linear component combinations have extensively been studied in the context of image statistics, the investigation of other combination rules has been limited to relatively small scale applications [17, 16, 35, 19]. Applying a novel training scheme, we could overcome this limitation in the case of the MCA generative model. As with linear approaches, we found that Gabor-like basis functions are obtained. The statistics of their shapes, a subject that is currently and actively discussed [31, 10, 32, 26, 27], is markedly different, however. Future work should, thus, at least be aware that a linear combination of components is not the only possible choice. To recover the generating causes of image patches, a linear combination might, furthermore, not be the best choice. With the results presented in this work, it can neither be considered as the only practical one anymore.

**Acknowledgements.** We gratefully acknowledge funding by the German Federal Ministry of Education and Research (BMBF) in the project 01GQ0840 (BFNT Frankfurt) and by the German Research Foundation (DFG) in the project LU 1196/4-1. Furthermore, we gratefully acknowledge support by the Frankfurt Center for Scientific Computing (CSC Frankfurt) and thank Marc Henniges for his help with Fig. 2.

## Footnotes

[1]Filter parameters were chosen as in [27]; before the brightest 2% of the pixels were clamped to the maximal value of the remaining 98% (influence of light-reflections were reduced in this way).

# References

[1] B. A. Olshausen, D. J. Field. Emergence of simple-cell receptive field properties by learning a sparse code for natural images. *Nature*, 381:607 – 609, 1996.

[2] P. Comon. Independent component analysis, a new concept? *Signal Proc*, 36(3):287–314, 1994.

[3] A. J. Bell, T. J. Sejnowski. The "independent components" of natural scenes are edge filters. *Vision Research*, 37(23):3327 – 38, 1997.

[4] A. Hyvärinen, E. Oja. A fast fixed-point algorithm for independent component analysis. *Neural Computation*, 9(7):1483–1492, 1997.

[5] H. Lee, A. Battle, R. Raina, A. Ng. Efficient sparse coding algorithms. *NIPS 22*, 801–808, 2007.

[6] M. W. Seeger. Bayesian Inference and Optimal Design for the Sparse Linear Model. *Journal of Machine Learning Research*, 759–813, 2008.

[7] P. Dayan, L. F. Abbott. *Theoretical Neuroscience*. MIT Press, Cambridge, 2001.

[8] P. Berkes, R. Turner, M. Sahani. On sparsity and overcompleteness in image models. *NIPS 20*, 2008.

[9] B. A. Olshausen, K. J. Millman. Learning sparse codes with a mixture-of-Gaussians prior. *NIPS 12*, 841–847, 2000.

[10] M. Rehn, F. T. Sommer. A network that uses few active neurones to code visual input predicts the diverse shapes of cortical receptive fields. *J Comp Neurosci*, 22(2):135–146, 2007.

[11] A. Hyvärinen, P. Hoyer. Emergence of phase-and shift-invariant features by decomposition of natural images into independent feature subspaces. *Neural Computation*, 12(7):1705–1720, 2000.

[12] F. Sinz, E. P. Simoncelli, M. Bethge. Hierarchical modeling of local image features through Lp-nested symmetric distributions. *NIPS 22*, 1696–1704, 2009.

[13] D. Zoran, Y. Weiss. The "Tree-Dependent Components" of Natural Images are Edge Filters. *NIPS 22*, 2340–2348, 2009.

[14] B. S. Everitt. *An Introduction to Latent Variable Models*. Chapman and Hall, 1984.

[15] D. D. Lee, H. S. Seung. Learning the parts of objects by non-negative matrix factorization. *Nature*, 401(6755):788–91, 1999.

[16] P. Dayan, R. S. Zemel. Competition and multiple cause models. *Neural Computation*, 7:565-579, 1995.

[17] E. Saund. A multiple cause mixture model for unsupervised learning. *Neural Computation*, 7:51-71, 1995.

[18] H. Lappalainen, X. Giannakopoulos, A. Honkela, J. Karhunen. Nonlinear independent component analysis using ensemble learning: Experiments and discussion. *Proc. ICA*, 2000.

[19] J. Lücke, M. Sahani. Maximal causes for non-linear component extraction. *Journal of Machine Learning Research*, 9:1227 – 1267, 2008.

[20] N. Jojic, B. Frey. Learning flexible sprites in video layers. *CVPR*, 199–206, 2001.

[21] N. Le Roux, N. Heess, J. Shotton, J. Winn. Learning a generative model of images by factoring appearance and shape. Technical Report, Microsoft Research, 2010.

[22] R. Neal, G. Hinton. A view of the EM algorithm that justifies incremental, sparse, and other variants. M. I. Jordan, editor, *Learning in Graphical Models*. Kluwer, 1998.

[23] J. Lücke, R. Turner, M. Sahani, M. Henniges. Occlusive Components Analysis. *NIPS*, 1069-1077, 2009.

[24] P. O. Hoyer. Non-negative matrix factorization with sparseness constraints. *Journal of Machine Learning Research*, 5:1457–1469, 2004.

[25] J. P. Jones, L. A. Palmer. An evaluation of the two-dimensional gabor filter model of simple receptive fields in cat striate cortex. *Journal of Neurophysiology*, 58(6):1233 – 1258, 1987.

[26] P. Berkes, B.L. White, J. Fiser. No evidence for active sparsification in the visual cortex. *NIPS 22*, 2009.

[27] J. Lücke. Receptive field self-organization in a model of the fine-structure in V1 cortical columns. *Neural Computation*, 21(10):2805–2845, 2009.

[28] J. H. van Hateren, A. van der Schaaf. Independent component filters of natural images compared with simple cells in primary visual cortex. *Proc Roy Soc London B*, 265:359 – 366, 1998.

[29] D. C. Somers, S. B. Nelson, M. Sur. An emergent model of orientation selectivity in cat visual cortical simple cells. *The Journal of Neuroscience*, 15:5448 – 5465, 1995.

[30] J. Lücke. Learning of representations in a canonical model of cortical columns. *Cosyne 2006*, 100, 2006.

[31] S. Osindero, M. Welling, G. E. Hinton. Topographic product models applied to natural scene statistics. *Neural Computation*, 18:381 – 414, 2006.

[32] D. Arathorn, B. Olshausen, J. DiCarlo. Functional requirements of a visual theory. *Workshop Cosyne*. www.cosyne.org/c/index.php?title=Functional_requirements_of_a_visual_theory, 2007.

[33] D. L. Ringach. Spatial structure and symmetry of simple-cell receptive fields in macaque primary visual cortex. *Journal of Neurophysiology*, 88:455 – 463, 2002. Data retrieved 2006 from *manuelita.psych.ucla.edu/~dario*.

[34] B. A. Olshausen, D. J. Field. Sparse coding with an overcomplete basis set: A strategy employed by V1? *Vision Research*, 37(23):3311–3325, 1997.

[35] S. Denéve, T. Lochmann, U. Ernst. Spike based inference in a network with divisive inhibition. *NeuralComp*, Marseille, 2008.

